# ICA-Based Clustering of Genes from Microarray Expression Data

**Su-In Lee[*] and Serafim Batzoglou[§]**
[*]Department of Electrical Engineering
[§]Department of Computer Science
Stanford University, Stanford, CA 94305
*silee@stanford.edu, serafim@cs.stanford.edu*

## Abstract

We propose an unsupervised methodology using independent component analysis (ICA) to cluster genes from DNA microarray data. Based on an ICA mixture model of genomic expression patterns, linear and nonlinear ICA finds components that are specific to certain biological processes. Genes that exhibit significant up-regulation or down-regulation within each component are grouped into clusters. We test the statistical significance of enrichment of gene annotations within each cluster. ICA-based clustering outperformed other leading methods in constructing functionally coherent clusters on various datasets. This result supports our model of genomic expression data as composite effect of independent biological processes. Comparison of clustering performance among various ICA algorithms including a kernel-based nonlinear ICA algorithm shows that nonlinear ICA performed the best for small datasets and natural-gradient maximization-likelihood worked well for all the datasets.

## 1 Introduction

Microarray technology has enabled genome-wide expression profiling, promising to provide insight into underlying biological mechanism involved in gene regulation. To aid such discoveries, mathematical tools that are versatile enough to capture the underlying biology and simple enough to be applied efficiently on large datasets are needed. Analysis tools based on novel data mining techniques have been proposed [1]-[6]. When applying mathematical models and tools to microarray analysis, clustering genes that have the similar biological properties is an important step for three reasons: reduction of data complexity, prediction of gene function, and evaluation of the analysis approach by measuring the statistical significance of biological coherence of gene clusters.

Independent component analysis (ICA) linearly decomposes each of $N$ vectors into $M$ common component vectors ($N \geq M$) so that each component is statistically as independent from the others as possible. One of the main applications of ICA is blind

source separation (BSS) that aims to separate source signals from their mixtures. There have been a few attempts to apply ICA to the microarray expression data to extract meaningful signals each corresponding to independent biological process [5]-[6]. In this paper, we provide the first evidence that ICA is a superior mathematical model and clustering tool for microarray analysis, compared to the most widely used methods namely PCA and k-means clustering. We also introduce the application of nonlinear ICA to microarray analysis, and show that it outperforms linear ICA on some datasets.

We apply ICA to microarray data to decompose the input data into statistically independent components. Then, genes are clustered in an unsupervised fashion into non-mutually exclusive clusters. Each independent component is assigned a putative biological meaning based on functional annotations of genes that are predominant within the component. We systematically evaluate the clustering performance of several ICA algorithms on four expression datasets and show that ICA-based clustering is superior to other leading methods that have been applied to analyze the same datasets. We also proposed a kernel based nonlinear ICA algorithm for dealing with more realistic mixture model. Among the different linear ICA algorithms including six linear and one nonlinear ICA algorithm, the natural-gradient maximum-likelihood estimation method (NMLE) [7]-[8] performs well in all the datasets. Kernel-based nonlinear ICA method worked better for three small datasets.

## 2   Mathematical model of genome-wide expression

Several distinct biological processes take place simultaneously inside a cell; each biological process has its own expression program to up-regulate or down-regulate the level of expression of specific sets of genes. We model a genome-wide expression pattern in a given condition (measured by a microarray assay) as a mixture of signals generated by statistically independent biological processes with different activation levels. We design two kinds of models for genomic expression pattern: a linear and nonlinear mixture model.

Suppose that a cell is governed by $M$ independent biological processes $S = (s_1, ..., s_M)^T$, each of which is a vector of $K$ gene expression levels, and that we measure the levels of expression of all genes in $N$ conditions, resulting in a microarray expression matrix $X = (x_1, ..., x_N)^T$. The expression level at each different condition $j$ can be expressed as linear combinations of the $M$ biological processes: $x_j = a_{j1}s_1 + ... + a_{jM}s_M$. We can express this idea concisely in matrix notation as follows.

$$X = AS, \qquad \begin{bmatrix} x_1 \\ \vdots \\ x_N \end{bmatrix} = \begin{bmatrix} a_{11} & \cdots & a_{1M} \\ \vdots & & \vdots \\ a_{N1} & \cdots & a_{NM} \end{bmatrix} \begin{bmatrix} s_1 \\ \vdots \\ s_M \end{bmatrix} \qquad (1)$$

More generally, we can express $X = (x_1, ..., x_N)^T$ as a post-nonlinear mixture of the underlying independent processes as follows, where $f(.)$ is a nonlinear mapping from $N$ to $N$ dimensional space.

$$X = f(AS), \qquad \begin{bmatrix} x_1 \\ \vdots \\ x_N \end{bmatrix} = f\left( \begin{bmatrix} a_{11} & \cdots & a_{1M} \\ \vdots & & \vdots \\ a_{N1} & \cdots & a_{NM} \end{bmatrix} \begin{bmatrix} s_1 \\ \vdots \\ s_M \end{bmatrix} \right) \qquad (2)$$

# 3 Independent component analysis

In the models described above, since we assume that the underlying biological processes are independent, we suggest that vectors $S=(s_1,...,s_M)$ are statistically independent and so ICA can recover $S$ from the observed microarray data $X$. For linear ICA, we apply natural-gradient maximum estimation (NMLE) method which was proposed in [7] and was made more efficient by using natural gradient method in [8]. We also apply nonlinear ICA using reproducible kernel Hilbert spaces (RKHS) based on [9], as follows:

1. We map the $N$ dimensional input data $x_i$ to $\Phi(x_i)$ in the feature space by using the *kernel trick*. The feature space is defined by the relationship $\Phi(x_i)^T\Phi(x_j)=k(x_{i,}, x_j)$. That is, inner product of mapped data is determined to by a kernel function $k(.,.)$ in the input space; we used a Gaussian radial basis function (RBF) kernel $(k(x,y)=exp(-|x-y|^2))$ and a polynomial kernel of degree 2 $(k(x,y)=(x^Ty+1)^2)$. To perform mapping, we found orthonormal bases of the feature space by randomly sampling $L$ input data $v=\{v_1,...,v_L\}$ 1000 times and choosing one set minimizing the condition number of $\Phi_v=(\Phi(v_1),...,\Phi(v_L))$. Then, a set of orthonormal bases of the feature space is determined by the selected $L$ images of input data in $v$ as $\Xi = \Phi_v(\Phi_v^T\Phi_v)^{-1/2}$. We map all input data $x_1,...,x_K$, each corresponding to a gene, to $\Psi(x_1),...,\Psi(x_K)$ in the feature space with basis $\Xi$, as follows:

$$\Psi(x_i)=(\Phi_v^T\Phi_v)^{-1/2}\Phi_v^T\Phi_v(x_i)=\begin{bmatrix} k(v_1,v_1) & \cdots & k(v_1,v_L) \\ \vdots & & \vdots \\ k(v_L,v_1) & \cdots & k(v_L,v_L) \end{bmatrix}^{-1/2}\begin{bmatrix} k(v_1,x_i) \\ \vdots \\ k(v_L,x_i) \end{bmatrix}\in\Re^L \quad (1\leq i \leq K) \quad (3)$$

2. We linearly decompose the mapped data $\Psi=[\Psi(x_1),..,\Psi(x_K)]\in R^{L\times K}$ into statistically independent components using NMLE.

# 4 Proposed approach

The microarray dataset we are given is in matrix form where each element $x_{ij}$ corresponds to the level of expression of the *jth* gene in the *ith* experimental condition. Missing values are imputed by KNNImpute [10], an algorithm based on $k$ nearest neighbors that is widely used in microarray analysis. Given the expression matrix $X$ of $N$ experiment by $K$ genes, we perform the following steps.

1. Apply ICA to decompose $X$ into independent components $y_1,...,y_M$ as in Equations (1) and (2). Prior to applying ICA, remove any rows that make the expression matrix $X$ singular. After ICA, each component denoted by $y_i$ is a vector comprising $K$ loads gene expression levels, i.e., $y_i = (y_{i1},...,y_{iK})$. We chose to let the number of components $M$ to be maximized, which is equal the number of microarray experiments $N$ because the maximum for $N$ in our datasets was 250, which is smaller than the number of biological processes we hypothesize to act within a cell.

2. For each component, cluster genes according to their relative loads $y_{ij}/mean(y_i)$. Based on our ICA model, each component is a putative genomic expression program of an independent biological process. Thus, our hypothesis is that genes showing relatively high or low expression level within the component are the most important for the process. We create two clusters for each component: one cluster containing genes with expression level higher than a threshold, and one cluster containing genes with expression level lower than a threshold.

$$Cluster_{i,1} = \{gene\ j\ |\ y_{ij} > mean(\ y_i\ ) + c \times std(\ y_i\ )\}$$

$$Cluster_{i,2} = \{gene\ j\ |\ y_{ij} < mean(\ y_i\ ) - c \times std(\ y_i\ )\} \quad\quad (4)$$

Here, $mean(y_i)$ is the average, $std(y_i)$ is the standard deviation of $y_i$; and $c$ is an adjustable coefficient. The value of the coefficient $c$ was varied from 1.0 to 2.0 and the result for $c=1.25$ was presented in this paper. The results for other values of $c$ are similar, and are presented on the website www.stanford.edu/~silee/ICA/.

3. For each cluster, measure the enrichment of each cluster with genes of known functional annotations. Using the Gene Ontology (GO) [11] and KEGG [12] gene annotation databases, we calculate the $p$-value for each cluster with every gene annotation, which is the probability that the cluster contains the observed number of genes with the annotation by chance assuming the hypergeometric distribution (details in [4]). For each gene annotation, the minimum $p$-value that is smaller than $10^{-7}$ obtained from any cluster was collected. If no $p$-value smaller than $10^{-7}$ is found, we consider the gene annotation not to be detected by the approach. As a result, we can assign biological meaning to each cluster and the corresponding independent component and we can evaluate the clustering performance by comparing the collected minimum p-value for each gene annotation with that from other clustering approach.

## 5   Performance evaluation

We tested the ICA-based clustering to four expression datasets (D1—D4) described in Table 1.

Table 1: The four datasets used in our analysis

|  | ARRAY TYPE | DESCRIPTION | # OF GENES ($K$) | # OF EXPS ($N$) |
|---|---|---|---|---|
| D1 | Spotted | Budding yeast during cell cycle and CLB2/CLN3 overactive strain [13] | 4579 | 22 |
| D2 | Oligonucleotide | Budding yeast during cell cycle [14] | 6616 | 17 |
| D3 | Spotted | *C. elegans* in various conditions [3] | 17817 | 553 |
| D4 | Oligonucleotide | Normal human tissue including 19 kinds of tissues [15] | 7070 | 59 |

For D1 and D4, we compared the biological coherence of ICA components with that of PCA applied in the same datasets in [1] and [2], respectively. For D2 and D3, we compared with k-means clustering and the topomap method, applied in the same datasets in [4] and [3], respectively. We applied nonlinear ICA to D1, D2 and D4. Dataset D3 is very large and makes the nonlinear algorithm unstable.

D1 was preprocessed to contain log-ratios $x_{ij}=\log_2(R_{ij}/G_{ij})$ between red and green intensities. In [1], principal components, referred to as eigenarrays, were hypothesized to be genomic expression programs of distinct biological processes. We compared the biological coherence of independent components with that of principal components found by [1]. Comparison was done in two ways: (1) For each component, we grouped genes within top $x$% of significant up-regulation and down-regulation (as measured by the load of the gene in the component) into two clusters with $x$ adjusted from 5% to 45%. For each value of $x$, statistical significance was measured for clusters from independent components and compared with that from

principal components based on the minimum *p*-value for each gene annotation, as described in Section 4. We made a scatter plot to compare the negative log of the collected best *p*-values for each gene annotation when *x* is fixed to be 15%, shown in Figure 1 (a) (2) Same as before, except we did not fix the value of *x*; instead, we collected the minimum *p*-value from each method for each GO and KEGG gene annotation category and compared the collected *p*-values (Figure 1 (b)). For both cases, in the majority of the gene annotation categories ICA produced significantly lower *p*-values than PCA did, especially for gene annotation for which both ICA and PCA showed high significance.

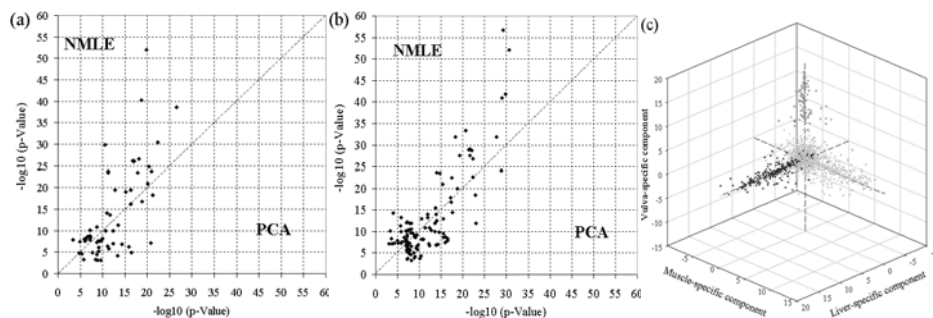

**Figure 1.** Comparison of linear ICA (NMLE) to PCA on dataset D1 (a) when x is fixed to be 15%; (b) when x is not fixed. (c) Three independent components of dataset D4. Each gene is mapped to a point based on the value assigned to the gene in three independent components, which are enriched with liver- (red), Muscle- (orange) and vulva-specific (green) genes, respectively.

The expression levels of genes in D4 were normalized across the 59 experiments, and the logarithms of the resulting values were taken. Experiments 57, 58, and 59 were removed because they made the expression matrix nearly singular. In [2], a clustering approach based on PCA and subsequent visual inspection was applied to an earlier version of this dataset, containing 50 of the 59 samples. After we performed ICA, the most significant independent components were enriched for liver-specific, muscle-specific and vulva-specific genes with *p*-value of $10^{-133}$, $10^{-124}$ and $100^{-117}$, respectively. In the ICA liver cluster, 198 genes were liver specific (out of a total of 244), as compared with the 23 liver-specific genes identified in [2] using PCA. The ICA muscle cluster of 235 genes contains 199 muscle specific genes compared to 19 muscle-specific genes identified in [2]. We generated a 3-dimensional scatter plot of the load expression levels of all genes annotated in [15] on these significant ICA components in Figure 1 (c). We can see that the liver-specific, muscle-specific and vulva-specific genes are strongly biased to lie on the x-, y-, and z- axis, respectively. We applied nonlinear ICA on this dataset and the first four most significant clusters from nonlinear ICA with Gaussian RBF kernel were muscle-specific, liver-specific, vulva-specific and brain-specific with *p*-value of $10^{-158}$, $10^{-127}$, $10^{-112}$ and $10^{-70}$, respectively, showing considerable improvement over the linear ICA clusters.

For D2, variance-normalization was applied to the 3000 most variant genes as in [4]. The 17th experiment, which made the expression matrix close to singular, was removed. We measured the statistical significance of clusters as described in Section 4 and compared the smallest *p*-value of each gene annotation from our approach to that from *k*-means clustering applied to the same dataset [4]. We made a scatter plot

for comparing the negative log of the smallest p-value (y-axis) from ICA clusters with that from $k$-means clustering (x-axis). The coefficient $c$ is varied from 1.0 to 2.0 and the superiority of ICA-based clustering to $k$-means clustering does not change. In many practical settings, estimation of the best $c$ is not needed; we can adjust $c$ to get a desired size of the cluster unless our focus is to blindly find the size of clusters. Figure 2 (a) (b) (c) shows for $c=1.25$ a comparison of the performance of linear ICA (NMLE), nonlinear ICA with Gaussian RBF kernel (NICA gauss), and $k$-means clustering ($k$-means).

For D3, first we removed experiments that contained more than 7000 missing values, because ICA does not perform properly when the dataset contains many missing values. The 250 remaining experiments were used, containing expression levels for 17817 genes preprocessed to be log-ratios $x_{ij}=\log_2(R_{ij}/G_{ij})$ between red and green intensities. We compared the biological coherence of clusters by our approach with that of topomap-based approach applied to the same dataset in [3]. The result when $c=1.25$ is plotted in the Figure 2 (d). We observe that the two methods perform very similarly, with most categories having roughly the same $p$-value in ICA and in the topomap clusters. The topomap clustering approach performs slightly better in a larger fraction of the categories. Still, we consider this performance a confirmation that ICA is a widely applicable method that requires minimal training: in this case the missing values and high diversity of the data make clustering especially challenging, while the topomap approach was specifically designed and manually trained for this dataset as described in [3].

Finally, we compared different ICA algorithms in terms of clustering performance. We tested six linear ICA methods: Natural Gradient Maximum Likelihood Estimation (NMLE) [7][8], Joint Approximate Diagonalization of Eigenmatrices [16], Fast Fixed Point ICA with three different measures of non-Gaussianity [17], and Extended Information Maximization (Infomax) [18]. We also tested two kernels for nonlinear ICA: Gaussian RBF kernel, and polynomial kernel (NICA ploy). For each dataset, we compared the biological coherence of clusters generated by each method. Among the six linear ICA algorithms, NMLE was the best in all datasets. Among both linear and nonlinear methods, the Gaussian kernel nonlinear ICA method was the best in Datasets D1, D2 and D4, the polynomial kernel nonlinear ICA method was best in Dataset D4, and NMLE was best in the large datasets (D3 and D4). In Figure 3, we compare the NMLE method with three other ICA methods for the dataset D2. Overall, the NMLE algorithm consistently performed well in all datasets. The nonlinear ICA algorithms performed best in the small datasets, but were unstable in the two largest datasets. More comparison results are demonstrated in the website www.stanford.edu/~silee/ICA/.

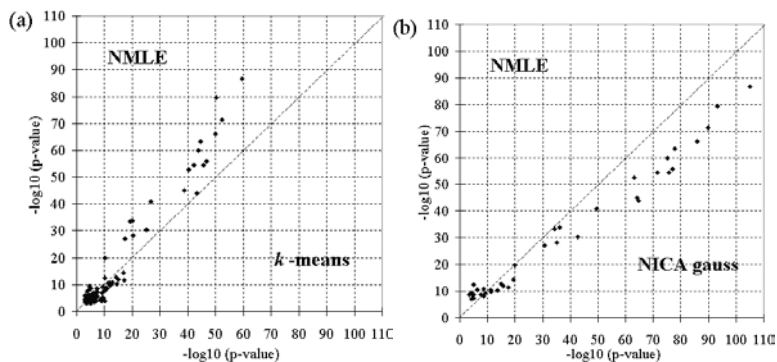

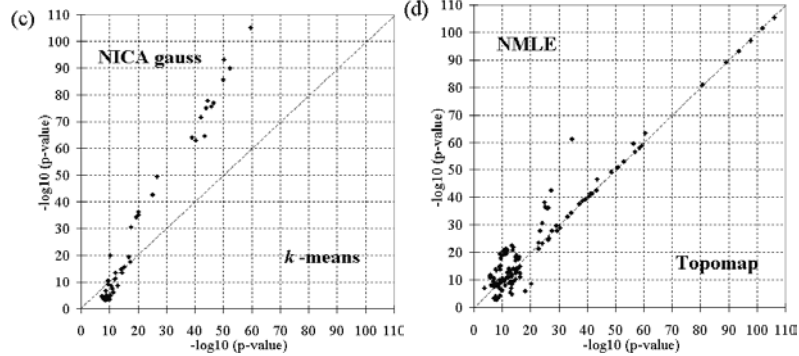

Figure 2: Comparison of (a) linear ICA (NMLE) with k-means clustering, (b) nonlinear ICA with Gaussian RBF kernel to linear ICA (NMLE), and (c) nonlinear ICA with Gaussian RBF kernel to k-means clustering on the dataset D2. (d) Comparison of linear ICA (NMLE) to topomap-based approach on the dataset D3.

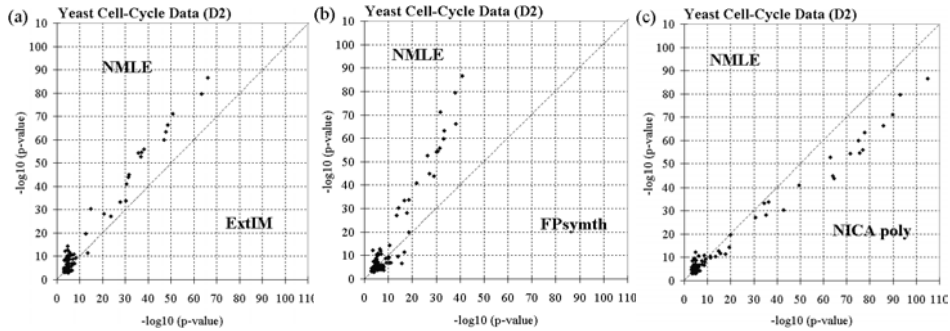

Figure 3: Comparison of linear ICA (NMLE) to (a) Extended Infomax ICA algorithm, (b) Fast ICA with symmetric orthogonalization and tanh nonlinearity and (c) Nonlinear ICA with polynomial kernel of degree 2 on the Dataset (B).

## 6   Discussion

ICA is a powerful statistical method for separating mixed independent signals. We proposed applying ICA to decompose microarray data into independent gene expression patterns of underlying biological processes, and to group genes into clusters that are mutually non-exclusive with statistically significant functional coherence. Our clustering method outperformed several leading methods on a variety of datasets, with the added advantage that it requires setting only one parameter, namely the fraction $c$ of standard deviations beyond which a gene is considered to be associated with a component's cluster. We observed that performance was not very sensitive to that parameter, suggesting that ICA is robust enough to be used for clustering with little human intervention.

The empirical performance of ICA in our tests supports the hypothesis that statistical independence is a good criterion for separating mixed biological signals in microarray data. The Extended Infomax ICA algorithm proposed in [18] can automatically determine whether the distribution of each source signal is super-Gaussian or sub-Gaussian. Interestingly, the application of Extended Infomax ICA to all the

expression datasets uncovered no source signal with sub-Gaussian distribution. A likely explanation is that global gene expression profiles are mixtures of super-Gaussian sources rather than of sub-Gaussian sources. This finding is consistent with the following intuition: underlying biological processes are super-Gaussian, because they affect sharply the relevant genes, typically a small fraction of all genes, and leave the majority of genes relatively unaffected.

## Acknowledgments

We thank Te-Won Lee for helpful feedback. We thank Relly Brandman, Chuong Do, and Yueyi Liu for edits to the manuscript.

## References

[1] Alter O, Brown PO, Botstein D. *Proc. Natl. Acad. Sci. USA* 97(18):10101-10106, 2000.

[2] Misra J, Schmitt W, et al. Genome Research 12:1112-1120, 2002.

[3] Kim SK, Lund J, et al. *Science* 293:2087-2092, 2001.

[4] Tavazoie S, Hughes JD, et al. *Nature Genetics* 22(3):281-285, 1999.

[5] Hori G, Inoue M, et al. Proc. 3[rd] Int. Workshop on Independent Component Analysis and Blind Signal Separation, Helsinki, Finland, pp. 151-155, 2000.

[6] Liebermeister W. *Bioinformatics* 18(1):51-60, 2002.

[7] Bell AJ. and Sejnowski TJ. *Neural Computation*, 7:1129-1159, 1995.

[8] Amari S, Cichocki A, et al. In *Advances in Neural Information Processing Systems* 8, pp. 757-763. Cambridge, MA: MIT Press, 1996.

[9] Harmeling S, Ziehe A, et al. In *Advances in Neural Information Processing Systems* 8, pp. 757-763. Cambridge, MA: MIT Press, .

[10] Troyanskaya O., Cantor M, et al. *Bioinformatics* 17:520-525, 2001.

[11] The Gene Ontology Consortium. *Genome Research* 11:1425-1433, 2001.

[12] Kanehisa M., Goto S. *In Current Topics in Computational Molecular Biology,* pp. 301–315. MIT-Press, Cambridge, MA, 2002.

[13] Spellman PT, Sherlock G, et al. *Mol. Biol. Cell* 9:3273-3297, 1998.

[14] Cho RJ, Campell MJ, et al. *Molecular Cell* 2:65-73, 1998.

[15] Hsiao L, Dangond F, et al. *Physiol. Genomics* **7:**97-104, 2001.

[16] Cardoso JF, *Neural Computation* **11(1)**:157-192, 1999.

[17] Hyvarinen A. *IEEE Transactions on Neural Network* **10(3):**626–634, 1999.

[18] Lee TW, Girolami M, et al. *Neural Computation* **11:**417–441, 1999.
